# Learning to Search Efficiently in High Dimensions

**Zhen Li** [*]
UIUC
zhenli3@uiuc.edu

**Huazhong Ning**
Google Inc.
huazhong@gooogle.com

**Liangliang Cao**
IBM T.J. Watson Research Center
liangliang.cao@us.ibm.com

**Tong Zhang**
Rutgers University
tzhang@stat.rutgers.edu

**Yihong Gong**
NEC China
ygongca@gmail.com

**Thomas S. Huang** [*]
UIUC
huang@ifp.uiuc.edu

## Abstract

High dimensional similarity search in large scale databases becomes an important challenge due to the advent of Internet. For such applications, specialized data structures are required to achieve computational efficiency. Traditional approaches relied on algorithmic constructions that are often data independent (such as Locality Sensitive Hashing) or weakly dependent (such as kd-trees, $k$-means trees). While supervised learning algorithms have been applied to related problems, those proposed in the literature mainly focused on learning hash codes optimized for compact embedding of the data rather than search efficiency. Consequently such an embedding has to be used with linear scan or another search algorithm. Hence *learning to hash* does not directly address the search efficiency issue. This paper considers a new framework that applies supervised learning to directly optimize a data structure that supports efficient large scale search. Our approach takes both search quality and computational cost into consideration. Specifically, we learn a *boosted search forest* that is optimized using pair-wise similarity labeled examples. The output of this search forest can be efficiently converted into an inverted indexing data structure, which can leverage modern text search infrastructure to achieve both scalability and efficiency. Experimental results show that our approach significantly outperforms the start-of-the-art learning to hash methods (such as spectral hashing), as well as state-of-the-art high dimensional search algorithms (such as LSH and $k$-means trees).

## 1 Introduction

The design of efficient algorithms for large scale similarity search (such as nearest neighbor search) has been a central problem in computer science. This problem becomes increasingly challenging in modern applications because the scale of modern databases has grown substantially and many of them are composed of high dimensional data. This means that classical algorithms such as kd-trees are no longer suitable [25] and new algorithms have to be designed to handle high dimensionality. However, existing approaches for large scale search in high dimension relied mainly on algorithmic constructions that are either data independent or weakly dependent. Motivated by the success of machine learning in the design of ranking functions for information retrieval (the *learning to rank* problem [13, 9]) and the design of compact embedding into binary codes (the *learning to hash* problem [10]), it is natural to ask whether we can use machine learning (in particular, supervised learning) to optimize data structures that can improve search efficiency. We call this problem **learning to search**, and this paper demonstrates that supervised learning can lead to improved search efficiency over algorithms that are not optimized using supervised information.

[*]These authors were sponsored in part by the U.S. National Science Foundation under grant IIS-1049332 EAGER and by the Beckman Seed Grant.

To leverage machine learning techniques, we need to consider a scalable search structure with parameters optimizable using labeled data. The data structured considered in this paper is motivated by the success of vocabulary tree method in image retrieval [18, 27, 15], which has been adopted in modern image search engines to find near duplicate images. Although the original proposal was based on "bag of local patch" image representation, this paper considers a general setting where each database item is represented as a high dimensional vector. Recent advances in computer vision show that it is desirable to represent images as numerical vectors of as high as thousands or even millions of dimensions [12, 28]. We can easily adapt the vocabulary tree to this setting: we partition the high dimensional space into disjoint regions using hierarchical $k$-means, and regard them as the "vocabulary". This representation can then be integrated into an inverted index based text search engine for efficient large scale retrieval. In this paper, we refer to this approach as $k$-means trees because the underlying algorithm is the same as in [5, 16]. Note that $k$-means trees can be used for high dimensional data, while the classical kd-trees [1, 3, 22] are limited to dimensions of no more than a few hundreds.

In this paper, we also adopt the tree structural representation, and propose a learning algorithm to construct the trees using supervised data. It is worth noting that the $k$-means trees approach suffers from several drawbacks that can be addressed in our approach. First the $k$-means trees only use unsupervised clustering algorithm, which is not optimized for search purposes; as we will show in the experiments, by employing supervised information, our learning to search approach can achieve significantly better performance. Second, the underlying $k$-means clustering limits the $k$-means tree approach to Euclidean similarity measures (though possible to extended to Bregman distances), while our approach can be easily applied to more general metrics (including semantic ones) that prove effective in many scenarios [8, 11, 7]. Nevertheless our experiments still focus on Euclidean distance search, which is to show the advantage over the $k$-means trees.

The learning to search framework proposed in this paper is based on a formulation of search as a supervised learning problem that jointly optimizes two key factors of search: *retrieval quality* and *computational cost*. Specifically, we learn a set of *selection functions* in the form of a tree ensemble, as motivated by the aforementioned kd-trees and $k$-means trees approaches. However, unlike the traditional methods that are based only on unsupervised information, our trees are learned under the supervision of pairwise similarity information, and are optimized for the defined search criteria, i.e., to maximize the retrieval quality while keeping the computational cost low. In order to form the forest, boosting is employed to learn the trees sequentially. We call this particular method **Boosted Search Forest** (BSF).

It is worth comparing the influential Locality Sensitive Hashing (LSH) [6, 2] approach with our learning to search approach. The idea of LSH is to employ random projections to approximate the Euclidean distance of original features. An inverted index structure can be constructed based on the hashing results [6], which facilitates efficient search. However, the LSH algorithm is completely data independent (using random projections), and thus the data structure is constructed without any learning. While interesting theoretical results can be obtained for LSH, as we shall see with the experiments, in practice its performance is inferior to the data-dependent search structures optimized via the learning to search approach of this paper.

Another closely related problem is *learning to hash*, which includes BoostSSC [20], Spectral Hashing [26], Restricted Boltzmann Machines [19], Semi-Supervised Hashing [24], Hashing with Graphs [14], etc. However, the motivation of hashing problem is fundamentally different from that of the search problem considered in this paper. Specifically, the goal of learning to hash is to embed data into compact binary codes so that the hamming distance between two codes reflects their original similarity. In order to perform efficient hamming distance search using the embedded representation, an additional efficient algorithmic structure is still needed. (How to come up with such an efficient algorithm is an issue usually ignored by learning to hash algorithms.) The compact hash codes were traditionally believed to achieve low search latency by employing either linear scan, hash table lookup, or more sophisticated search mechanism. As we shall see in our experiments, however, linear scan on the Hamming space is not a feasible solution for large scale search problems. Moreover, if other search data structure is implemented on top of the hash code, the optimality of the embedding is likely to be lost, which usually yields suboptimal solution inferior to directly optimizing a search criteria.

## 2 Background

Given a database $\mathcal{X} = \{x_1, \ldots, x_n\}$ and a query $q$, the search problem is to return top ranked items from the database that are most similar to the query. Let $s(q,x) \geq 0$ be a ranking function that measures the similarity between $q$ and $x$. In large-scale search applications, the database size $n$ can be billions or larger. Explicitly evaluating the ranking function $s(q,x)$ against all samples is very expensive. On the other hand, in order to achieve accurate search results, a complicated ranking function $s(q,x)$ is indispensible.

Modern search engines handle this problem by first employing a non-negative *selection function* $T(q,x)$ that selects a small set of candidates $\mathcal{X}_q = \{x : T(q,x) > 0, x \in \mathcal{X}\}$ with most of the top ranked items ($T(q,x) = 0$ means "not selected"). This is called **candidate selection** stage, which is followed by a **reranking** stage where a more costly ranking function $s(q,x)$ is evaluated on $\mathcal{X}_q$.

Two properties of the selection function $T(q,x)$ are: 1) It must be evaluated much more efficiently than the ranking function $s(q,x)$. In particular, for a given query, the complexity of evaluating $T(q,x)$ over the entire dataset should be sublinear or even constant, which is usually made possible by delicated data structures such as inverted index tables. 2) The selection function is an approximation to $s(q,x)$. In other word, with high probability, the more similar $q$ and $x$ are, the more likely $x$ is contained in $\mathcal{X}_q$ (which means $T(q,x)$ should take a larger value).

This paper focuses on the candidate selection stage, i.e., learning the selection function $T(q,x)$. In order to achieve both effectiveness and efficiency, three aspects need to be taken into account:

- $\mathcal{X}_q$ can be efficiently obtained (this is ensured by properties of selection function).
- The size of $\mathcal{X}_q$ should be small since it indicates the *computational cost* for reranking.
- The *retrieval quality* of $\mathcal{X}_q$ measured by the total similarity $\sum_{x \in \mathcal{X}_q} s(q,x)$ should be large.

Therefore, our objective is to retrieve a set of items that maximizes the ranking quality while lowering the computational cost (keeping the candidate set as small as possible). In additiona, to achieve search efficiency, the selection stage employs the inverted index structure as in standard text search engines to handle web-scale dataset.

## 3 Learning to Search

This section presents the proposed *learning to search* framework. We present the general formulation first, followed by a specific algorithm based on boosted search trees.

### 3.1 Problem Formulation

As stated in Section 2, the set of candidates returned for a query $q$ is given by $\mathcal{X}_q = \{x \in \mathcal{X} : T(q,x) > 0\}$. Intuitively, the *quality* of this candidate set can be measured by the overall similarities while the reranking cost is linear in $|\mathcal{X}_q|$. Mathematically, we define:

$$\textbf{Retrieval Quality:} \quad Q(T) = \sum_q \sum_{x \in \mathcal{X}} s(q,x)\mathbf{1}(T(q,x) > 0) \tag{1}$$

$$\textbf{Computational Cost:} \quad C(T) = \sum_q \sum_{x \in \mathcal{X}} \mathbf{1}(T(q,x) > 0) \tag{2}$$

where $\mathbf{1}(\cdot)$ is the indicator function.

The **learning to search** framework considers the search problem as a machine learning problem that finds the optimal selection function $T$ as follows:

$$\max_T Q(T) \quad \text{subject to } C(T) \leq C_0, \tag{3}$$

where $C_0$ is the upper-bound of computational cost. Alternatively, we can rewrite the optimization problem in (3) by applying Lagrange multiplier:

$$\max_T Q(T) - \lambda C(T), \tag{4}$$

where $\lambda$ is a tuning parameter that balances the retrieval quality and computational cost.

To simplify the learning process, we assume that the queries are randomly drawn from the database. Let $x_i$ and $x_j$ be two arbitrary samples in the dataset, and let $s_{ij} = s(x_i, x_j) \in \{1, 0\}$ indicate if they are "similar" or "dissimilar". Problem in (4) becomes:

$$
\begin{aligned}
\max_T J(T) \;&=\; \max_T \sum_{i,j} s_{ij} \mathbf{1}(T(x_i, x_j) > 0) - \lambda \sum_{i,j} \mathbf{1}(T(x_i, x_j) > 0) \\
&=\; \max_T \sum_{i,j} z_{ij} \mathbf{1}(T(x_i, x_j) > 0)
\end{aligned}
\tag{5}
$$

where

$$
z_{ij} = \begin{cases} 1 - \lambda & \text{for similar pairs} \\ -\lambda & \text{for dissimilar pairs} \end{cases}.
\tag{6}
$$

## 3.2 Learning Ensemble Selection Function via Boosting

Note that (5) is nonconvex in $T$ and thus is difficult to optimize. Inspired by AdaBoost [4], we employ the standard trick of using a convex relaxation, and in particular, we consider the exponential loss as a convex surrogate:

$$
\min_T \mathcal{L}(T) = \sum_{i,j} e^{-z_{ij} T(x_i, x_j)} = \mathbb{E}[e^{-z T(x_i, x_j)}].
\tag{7}
$$

Here we replace the summation over $\forall (x_i, x_j) \in \mathcal{X} \times \mathcal{X}$ by the expectation over two *i.i.d.* random variables $x_i$ and $x_j$. We also drop the subscripts of $z_{ij}$ and regard $z$ as a random variable conditioned on $x_i$ and $x_j$.

We define the ensemble selection function as a weighted sum of a set of base selection functions:

$$
T(x, y) = \sum_{m=1}^{M} c_m \cdot t_m(x_i, x_j).
\tag{8}
$$

Suppose we have learnt $M$ base functions, and we are about to learn the $(M + 1)$-th selection function, denoted as $t(x_i, x_j)$ with weight given by $c$. The updated loss function is hence given by

$$
\min_t \mathcal{L}(t, c) = \mathbb{E}[e^{-z[T(x_i, x_j) + ct(x_i, x_j)]}] = \mathbb{E}_w[e^{-czt(x_i, x_j)}],
\tag{9}
$$

where $\mathbb{E}_w[\cdot]$ denotes the *weighted expectation* with weights given by

$$
w_{ij} = w(x_i, x_j) = e^{-z T(x_i, x_j)} = \begin{cases} e^{-(1-\lambda)T(x_i, x_j)} & \text{for similar pairs} \\ e^{\lambda T(x_i, x_j)} & \text{for dissimilar pairs} \end{cases}
\tag{10}
$$

This reweighting scheme leads to the boosting algorithm in Algorithm 1.

In many application scenarios, each base selection function $t(x_i, x_j)$ takes only binary values 1 or 0. Thus, we may want to minimize $\mathcal{L}(t, c)$ by choosing the optimal value of $t(x_i, x_j)$ for any given pair $(x_i, x_j)$.

**Case 1:** $t(x_i, x_j) = 0$

$$
\mathcal{L}(t, c) = \mathbb{E}_w[e^{-0}] = 1.
\tag{11}
$$

**Case 2:** $t(x_i, x_j) = 1$

$$
\mathcal{L}(t, c) = \mathbb{E}_w[e^{-zc}] = e^{-(1-\lambda)c} \cdot \mathbf{P}_w[s_{ij} = 1 | x_i, x_j] + e^{\lambda c} \cdot \mathbf{P}_w[s_{ij} = 0 | x_i, x_j].
\tag{12}
$$

Comparing the two cases leads to:

$$
t^*(x_i, x_j) = \begin{cases} 1 & \text{if } \mathbf{P}_w[s_{ij} = 1 | x_i, x_j] > \frac{1 - e^{-\lambda c}}{1 - e^{-c}} \\ 0 & \text{otherwise} \end{cases}
\tag{13}
$$

To find the optimal $c$, we first decompose $\mathcal{L}$ in the following way:

$$
\begin{aligned}
\mathcal{L}(t, c) \;&=\; \mathbb{E}_w[e^{-czt(x_i, x_j)}] \\
&=\; \mathbf{P}_w[t(x_i, x_j) = 0 | x_i, x_j] + e^{-c(1-\lambda)} \cdot \mathbf{P}_w[t(x_i, x_j) = 1, s_{ij} = 1 | x_i, x_j] \\
&\quad + e^{c\lambda} \cdot \mathbf{P}_w[t(x_i, x_j) = 1, s_{ij} = 0 | x_i, x_j].
\end{aligned}
\tag{14}
$$

Taking the derivative of $\mathcal{L}$ with respect to $c$, we arrive at the optimal solution for $c$:

$$c^* = \log \frac{(1-\lambda)\mathbf{P}_w[t(x_i,x_j)=1, s_{ij}=1|x_i,x_j]}{\lambda \mathbf{P}_w[t(x_i,x_j)=0, s_{ij}=1|x_i,x_j]}. \tag{15}$$

---

**Algorithm 1** Boosted Selection Function Learning

---

**Input:** A set of data points $\mathcal{X}$; pairwise similarities $s_{ij} \in \{0,1\}$ and weights $w_{ij} = 1$
1: **for** $m \in 1,2,\cdots,M$ **do**
2:     Learn a base selection function $t_m(x,y)$ based on weights $w_{ij}$
3:     Update ensemble: $T(x_i,x_j) \leftarrow T(x_i,x_j) + c_m \cdot t_m(x_i,x_j)$
4:     Update weights:  $w_{ij} \leftarrow w_{ij} \cdot e^{-c_m \cdot t_m(x_i,x_j)}$
5: **end for**

---

### 3.3   Tree Implementation of Base Selection Function

Simultaneously solving (13) and (15) leads to the optimal solutions at each iteration of boosting. In practice, however, the optimality can hardly be achieved. This is particularly because the binary-valued base selection functions $t(x_i,x_j)$ has to be selected from limited function families to ensure the wearability (finite model complexity) and more importantly, the efficiency. As mentioned in Section 2, evaluating $t(q,x)$ for $\forall x \in \mathcal{X}$ needs to be accomplished in sublinear or constant time when a query $q$ comes. This suggests using an inverted table data structure as an efficient implantation of the selection function. Specifically, $t(x_i,x_j) = 1$ if $x_i$ and $x_j$ get hashed into the same bucket of the inverted table, and 0 otherwise. This paper considers trees (we name it "search trees") as an approximation to the optimal selection functions, and quick inverted table lookup follows naturally.

A natural consideration for the tree construction is that the tree must be balanced. However, we do not need to explicitly enforce this constraint: the balanceness is automatically favored by the term $C$ in (4) as balanced trees give the minimum computational cost. In this sense, unlike other methods that explicitly enforce balancing constraint, we relax it while jointly optimizing the retrieval quality and computational cost.

Consider a search tree with $L$ leaf nodes $\{\ell_1,\cdots,\ell_L\}$. The selection function given by this tree is defined as

$$t(x_i,x_j) = \sum_{k=1}^{L} t(x_i,x_j;\ell_k), \tag{16}$$

where $t(x_i,x_j;\ell_k) \in \{0,1\}$ indicating whether both $x_i$ and $x_j$ reach the same leaf node $\ell_k$. Similar to (5), the objective function for a search tree can be written as:

$$\max_t J = \max_t \sum_{i,j} w_{ij} z_{ij} \sum_{k=1}^{L} t(x_i,x_j;\ell_k) = \max_t \sum_{k=1}^{L} J^k, \tag{17}$$

where $J^k = \sum_{ij} w_{ij} z_{ij} t(x_i,x_j;\ell_k)$ is a partial objective function for the $k$-th leaf node, and $w_{ij}$ is given by (10).

The appealing additive property of the objective function $J$ makes it trackable to analyze each split when the search tree grows. In particular, we split the $k$-th leaf node into two child nodes $k(1)$ and $k(2)$ if and only if it increases the overall objective function $J^{k(1)} + J^{k(2)} > J^k$. Moreover, we optimize each split by choosing the one that maximizes $J^{k(1)} + J^{k(2)}$.

To find the optimal split for a leaf node $\ell_k$, we confine to the hyperplane split cases, i.e., a sample $x$ is assigned to the left child $\ell_{k(1)}$ if $p^\top x + b = \tilde{p}^\top \tilde{x} > 0$ and right child otherwise, where $\tilde{p} = [p^\top\ b]^\top$ and $\tilde{x} = [x^\top\ 1]^\top$ are the augmented projection and data vectors. The splitting criterion is given by:

$$\begin{aligned}
\max J^{k(1)} + J^{k(2)} &= \max_{\|\tilde{p}\|=1} \sum_{ij} w_{ij} z_{ij} \mathbf{1}(\tilde{p}^\top \tilde{x}_i \cdot \tilde{p}^\top \tilde{x}_j > 0) \\
&\approx \max_{\|\tilde{p}\|=1} \sum_{ij} w_{ij} z_{ij} [\tilde{p}^\top \tilde{x}_i \tilde{x}_j^\top \tilde{p}] \\
&= \max_{\|\tilde{p}\|=1} \tilde{p}^\top \tilde{X} M \tilde{X}^\top \tilde{p}, \tag{18}
\end{aligned}$$

where $M_{ij} = w_{ij}z_{ij}$, and $\tilde{X}$ is the stack of all augmented samples at node $\ell_k$. Note that as $\mathbf{1}(a > 0) = \frac{1}{2}\text{sign}(a) + \frac{1}{2}$ is non-differentiable, we approximate it using $\frac{1}{2}a + \frac{1}{2}$. The optimal $\tilde{p}$ of the above objective function is the eigenvector corresponding to the largest eigenvalue of $\tilde{X}M\tilde{X}^\top$.

The search tree construction algorithm is listed in Algorithm 2. In the implementation, if computation resource is critical, we may use stump functions to split the nodes with a large amount of samples, while applying the optimal projection $p$ to the small nodes. The selection of the stump functions is similar to that in traditional decision trees: on the given leaf node, a set of stump functions are attempted and the one that maximizes (17) is selected if the objective function increases.

---

**Algorithm 2** Search Tree Construction

---

**Input:** A set of data points $\mathcal{X}$; pairwise similarities $s_{ij} \in \{0,1\}$ and weights $w_{ij}$ given by (10)
**Output:** Tree $t$
 1: Assign $\mathcal{X}$ as root; enqueue root
 2: **repeat**
 3:     Find a leaf node $\ell$ in the queue; dequeue $\ell$
 4:     Find the optimal split for $\ell$ by solving (18)
 5:     **if** criteria in (17) increases **then**
 6:         Split $\ell$ into $\ell_1$ and $\ell_2$; enqueue $\ell_1$ and $\ell_2$
 7:     **end if**
 8: **until** Queue is empty

---

### 3.4 Boosted Search Forest

In summary, we present a Boosted Search Forest (BSF) algorithm to the *learning to search* problem. In the learning stage, this algorithm follows the boosting framework described in Algorithm 1 to learn an ensemble of selection functions; each base selection function, in the form of a search tree, is learned with Algorithm 2. We then build inverted indices by passing all data points through the learned search trees. In analogy to text search, each leaf node corresponds to an "index word" in the vocabulary and the data points reaching this leaf node are the "documents" associated with this "index word". In the candidate selection stage, instead of exhaustively evaluating $T(q, x)$ for $\forall x \in \mathcal{X}$, we only need to traverse the search trees and retrieve all items that collide with the query example for at least one tree. The selected candidate set, given by $\mathcal{X}_q = \{x \in \mathcal{X} : T(q, x) > 0\}$, is statistically optimized to have a small size (small computation cost) while containing a large number of relevant samples (good retrieval quality).

## 4 Experiments

We evaluate the Boosted Search Forest (BSF) algorithm on several image search tasks. Although a more general similarity measure can be used, for simplicity we set $s(x_i, x_j) \in \{0,1\}$ according to whether $x_j$ is within the top $K$ nearest neighbors ($K$-NN) of $x_i$ on the designated metric space. We use $K = 100$ in the implementation.

We compare the performance of BSF to two most popular algorithms on high dimensional image search: $k$-means trees and LSH. We also compare to a representative method in the learning to hash community: spectral hashing, although this algorithm was designed for Hamming embedding instead of search. Here linear scan is adopted on top of spectral hashing for search, because its more efficient alternatives are either directly compared (such as LSH) or can easily fail as noticed in [24]. Our experiment shows that exhaustive linear scan is not scalable, especially with long hash codes needed for better retrieval accuracy (see Table 1).

The above algorithms are most representative. We do not compare with other algorithms for several reasons. Fist, LSH was reported to be superior to kd-trees [21] and spectral hashing was reported to out-perform RBM and BoostSCC [26]. Second, kd-trees and its extensions still work on low dimensions, and is known to behave poorly on high dimension data like in image search. Third, since this paper focuses on learning to search, not learning to hash (Hamming embedding) or learning distance metrics that consider different goals, it is not essential to compare with more recent work on those topics such as [8, 11, 24, 7].

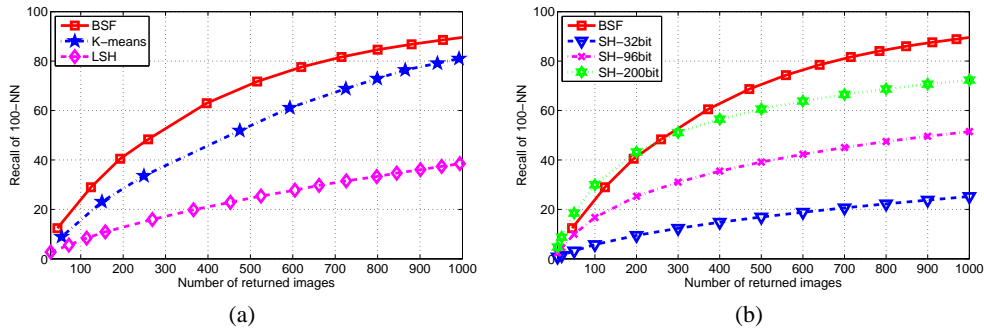

Figure 1: Comparison of Boosted Search Forest (BSF) on Concept1000 dataset with (a) $k$-means trees and LSH (b) Spectral Hashing (SH) of varying bits.

## 4.1 Concept-1000 Dataset

This dataset consists of more than 150K images of 1000 concepts selected from the Large Scale Concept Ontology for Multimedia (LSCOM) [17]. The LSCOM categories were specifically selected for multimedia annotation and retrieval, and have been used in the TRECVID video retrieval series. These concept names were inputed as queries in Google and Bing, and the top returned images were collected.

We choose the image representation proposed in [28], which is a high dimensional ($\sim$84K) feature with reported state-of-the-art performance in many visual recognition tasks. PCA is applied to reduce the dimension to 1000. We then randomly select around 6000 images as queries, and use the remaining ($\sim$150K) images as the search database.

In image search, we are interested in the overall quality of the set of candidate images returned by a search algorithm. This notion coincides with our formulation of the search problem in (4) that is aimed at maximizing retrieval quality while maintaining a relative low computational cost (for reranking stage). The number of returned images clearly reflects the computational cost, and the retrieval quality is measured by the recall of retrieved images, i.e., the number of retrieved images that are among the 100-NN of the query. Note that we use recall instead of accuracy because recall gives the upper-bound performance of the reranking stage.

Figure 1(a) shows the performance comparison with two search algorithms: $k$-means trees and LSH. Since our boosted search forest consists of tree ensembles, for a fair comparison, we also construct equivalent number of $k$-means trees (with random initializations) and multiple sets of LSH codes. Our proposed approach significantly outperforms $k$-means trees and LSH. The better performance is due to our learning to search formulation that simultaneously maximizes recall while minimizing the size of returned candidate set. In contrast, $k$-means trees uses only unsupervised clustering algorithm and LSH employs purely random projections. Moreover, the performance of $k$-means algorithm deteriorates when dimension increases.

It is still interesting to compare to spectral hashing, although it is not a search algorithm. Since our approach requires more trees when the number of returns increases, we implement spectral hashing with varying bits: 32-bit, 96-bit, and 200-bit. As illustrated in Figure 1(b), our approach significantly outperforms spectral hashing under all configurations. Although the search forest does not have an explicit concept of bits, we can measure it from the information theoretical point of view, by counting every binary-branching in the trees as one bit. In the experiment, our approach retrieves about 70% of 100-NN out of 500 returned images, after traversing 17 trees, each of 12 layers. This is equivalent to $17 \times 12 = 204$ bits. With the same number of bits, spectral hashing only achieves a recall rate around 60%.

## 4.2 One Million Tiny Images

In order to examine the scalability of BSF, we conducted experiments on a much larger database. We randomly sample one million images from the 80 Millions Tiny Images dataset [23] as the search database, and 5000 additional images as queries. We use the 384-dimensional GIST feature provided by the authors of [23]. Comparison with search algorithms (Figure 2(a)) and hashing methods

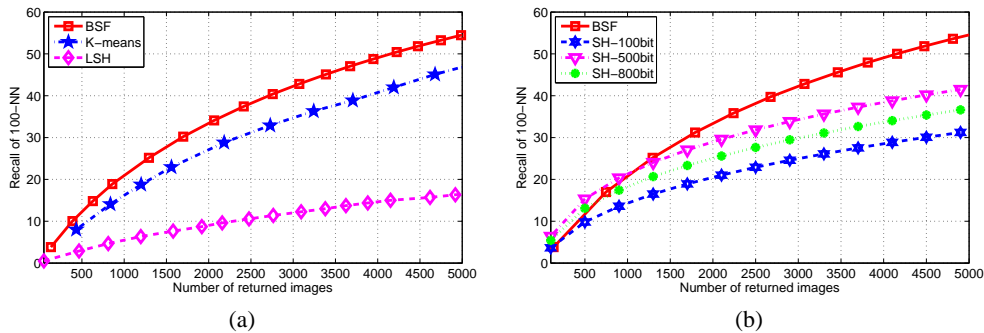

|     | (a) |     | (b) |
|-----|-----|-----|-----|

Figure 2: Comparison of Boosted Search Forest (BSF) on 1 Millions Tiny Images dataset with (a) $K$-means trees and LSH (b) Spectral Hashing (SH) of varying bits.

Table 1: Comparison of retrieval time in a database with 0.5 billion synthesized samples.

| #bits | 32 | 64 | 128 | 256 | 512 |
|-------|-----|-----|------|------|------|
| **Linear scan** | 1.55s | 2.74s | 5.13s | 10.11s | 19.79s |
| **Boosted search forest** | 0.006s | 0.009s | 0.017s | 0.034s | 0.073s |

(Figure 2(b)) are made in a similar way as in the previous section. Again, the BSF algorithm substantially outperforms the other methods: using 60 trees (less than 800 bits), our approach retrieves 55.0% of the 100-NN with 5000 returns (0.5% of the entire database), while $k$-means trees achieves only 47.1% recall rate and LSH and spectral hashing are even worse. Note that using more bits in spectral hashing can even hurt performance on this dataset.

## 4.3   Search Speed

All three aforementioned search algorithms (boosted search trees, $k$-means trees, and LSH) can naturally utilize inverted index structures to facilitate very efficient search. In particular, both our boosted search trees and $k$-means trees use the leaf nodes as the keys to index a list of data points in the database, while LSH uses multiple independently generated bits to form the indexing key. In this sense, all three algorithm has the same order of efficiency (constant time complexity).

On the other hand, in order to perform search with compact hamming codes generated by a learning to hash method (e.g. spectral hashing), one has to either use a linear scan approach or a hash table lookup technique that finds the samples within a radius-1 Hamming ball (or more complex methods like LSH). Although much more efficient, the hash table lookup approach is likely to fail as the dimension of hash code grows to a few dozens, as observed in [24]. The retrieval speed using exhaustive linear scan is, however, far from satisfactory. Table 1 clearly illustrates this phenomenon on a database of 0.5 billion synthesized items. Even small codes with 32 bits take around 1.55 seconds (without sorting). When the hash codes grow to 512 bits (which is not unusual for high-dimensional image/video data), the query time is almost 20 seconds. This is not acceptable for most real applications. On the contrary, our boosted search forest with 32 16-layer trees ($\sim$512 bits) responds in less than 0.073s. Our timing is carried out on a Intel Xeon Quad X5560 CPU, with a highly optimized implementation of Hamming distance which is at least 8–10 times faster than a naive implementation.

## 5   Conclusion

This paper introduces a learning to search framework for scalable similarity search in high dimensions. Unlike previous methods, our algorithm learns a boosted search forest by jointly optimizing search quality versus computational efficiency, under the supervision of pair-wise similarity labels. With a natural integration of the inverted index search structure, our method can handle web-scale datasets efficiently. Experiments show that our approach leads to better retrieval accuracy than the state-of-the-art search methods such as locality sensitive hashing and $k$-means trees.

# References

[1] J. S. Beis and D. G. Lowe. Shape indexing using approximate nearest-neighbour search in high-dimensional spaces. In *CVPR*, pages 1000–1006, 1997.

[2] M. Datar, N. Immorlica, P. Indyk, and V. S. Mirrokni. Locality-sensitive hashing scheme based on p-stable distributions. In *Symposium on Computational Geometry*, pages 253–262, 2004.

[3] J. Friedman, J. Bentley, and R. Finkel. An algorithm for finding best matches in logarithmic expected time. *ACM Transactions on Mathematical Software (TOMS)*, 3(3):209–226, 1977.

[4] J. Friedman, T. Hastie, and R. Tibshirani. Additive logistic regression: A statistical view of boosting. *The Annals of Statistics*, 28(2):337–374, 2000.

[5] K. Fukunage and P. Narendra. A branch and bound algorithm for computing k-nearest neighbors. *IEEE Transactions on Computers*, 100(7):750–753, 1975.

[6] A. Gionis, P. Indyk, and R. Motwani. Similarity search in high dimensions via hashing. In *VLDB*, pages 518–529, 1999.

[7] J. He, W. Liu, and S.-F. Chang. Scalable similarity search with optimized kernel hashing. In *KDD*, 2010.

[8] P. Jain, B. Kulis, and K. Grauman. Fast image search for learned metrics. In *CVPR*, 2008.

[9] V. Jain and M. Varma. Learning to re-rank: query-dependent image re-ranking using click data. In *WWW*, pages 277–286, 2011.

[10] B. Kulis and T. Darrell. Learning to hash with binary reconstructive embeddings. *NIPS*, 2009.

[11] B. Kulis and K. Grauman. Kernelized locality-sensitive hashing for scalable image search. In *ICCV*, 2009.

[12] Y. Lin, F. Lv, S. Zhu, M. Yang, T. Cour, K. Yu, L. Cao, and T. Huang. Large-scale image classification: fast feature extraction and svm training. In *CVPR*, 2011.

[13] T.-Y. Liu. Learning to rank for information retrieval. In *SIGIR*, page 904, 2010.

[14] W. Liu, J. Wang, S. Kumar, and S. Chang. Hashing with graphs. In *ICML*, 2011.

[15] F. Moosmann, B. Triggs, and F. Jurie. Fast discriminative visual codebooks using randomized clustering forests. In *NIPS*, pages 985–992, 2006.

[16] M. Muja and D. G. Lowe. Fast approximate nearest neighbors with automatic algorithm configuration. In *VISSAPP*, 2009.

[17] M. Naphade, J. Smith, J. Tesic, S. Chang, W. Hsu, L. Kennedy, A. Hauptmann, and J. Curtis. Large-scale concept ontology for multimedia. *IEEE Multimedia Magazine*, 13(3):86–91, 2006.

[18] D. Nistér and H. Stewénius. Scalable recognition with a vocabulary tree. In *CVPR*, pages 2161–2168, 2006.

[19] R. Salakhutdinov and G. E. Hinton. Semantic hashing. *Int. J. Approx. Reasoning*, 50(7):969–978, 2009.

[20] G. Shakhnarovich. *Learning task-specific similarity*. PhD thesis, Massachusetts Institute of Technology, 2005.

[21] G. Shakhnarovich, T. Darrell, and P. Indyk. *Nearest-Neighbor Methods in Learning and Vision: Theory and Practice*. The MIT Press, 2006.

[22] C. Silpa-Anan and R. Hartley. Optimised kd-trees for fast image descriptor matching. In *CVPR*, 2008.

[23] A. Torralba, R. Fergus, and W. T. Freeman. 80 million tiny images: A large data set for nonparametric object and scene recognition. *IEEE Trans. PAMI*, 30(11), 2008.

[24] J. Wang, O. Kumar, and S.-F. Chang. Semi-supervised hashing for scalable image retrieval. In *CVPR*, 2010.

[25] Weber, Roger, Schek, Hans J., and Blott, Stephen. A Quantitative Analysis and Performance Study for Similarity-Search Methods in High-Dimensional Spaces. In *VLDB*, 1998.

[26] Y. Weiss, A. Torralba, and R. Fergus. Spectral hashing. In *NIPS*, 2008.

[27] T. Yeh, J. J. Lee, and T. Darrell. Adaptive vocabulary forests br dynamic indexing and category learning. In *ICCV*, pages 1–8, 2007.

[28] X. Zhou, N. Cui, Z. Li, F. Liang, and T. S. Huang. Hierarchical gaussianization for image classification. In *ICCV*, 2009.

